# Compositionality of optimal control laws

**Emanuel Todorov**
Applied Mathematics and Computer Science & Engineering
University of Washington
todorov@cs.washington.edu

## Abstract

We present a theory of compositionality in stochastic optimal control, showing how task-optimal controllers can be constructed from certain primitives. The primitives are themselves feedback controllers pursuing their own agendas. They are mixed in proportion to how much progress they are making towards their agendas and how compatible their agendas are with the present task. The resulting composite control law is provably optimal when the problem belongs to a certain class. This class is rather general and yet has a number of unique properties – one of which is that the Bellman equation can be made linear even for non-linear or discrete dynamics. This gives rise to the compositionality developed here. In the special case of linear dynamics and Gaussian noise our framework yields analytical solutions (i.e. non-linear mixtures of LQG controllers) without requiring the final cost to be quadratic. More generally, a natural set of control primitives can be constructed by applying SVD to Green's function of the Bellman equation. We illustrate the theory in the context of human arm movements. The ideas of optimality and compositionality are both very prominent in the field of motor control, yet they have been difficult to reconcile. Our work makes this possible.

## 1 Introduction

Stochastic optimal control is of interest in many fields of science and engineering, however it remains hard to solve. Dynamic programming [1] and reinforcement learning [2] work well in discrete state spaces of reasonable size, but cannot handle continuous high-dimensional state spaces characteristic of complex dynamical systems. A variety of function approximation methods are available [3, 4], yet the shortage of convincing results on challenging problems suggests that existing approximation methods do not scale as well as one would like. Thus there is need for more efficient methods. The idea we pursue in this paper is compositionality. With few exceptions [5, 6] this good-in-general idea is rarely used in optimal control, because it is unclear what/how can be composed in a way that guarantees optimality of the resulting control law.

Our second motivation is understanding how the brain controls movement. Since the brain remains pretty much the only system capable of solving truly complex control problems, sensorimotor neuroscience is a natural (albeit under-exploited) source of inspiration. To be sure, a satisfactory understanding of the neural control of movement is nowhere in sight. Yet there exist theoretical ideas backed by experimental data which shed light on the underlying computational principles. One such idea is that biological movements are near-optimal [7, 8]. This is not surprising given that motor behavior is shaped by the processes of evolution, development, learning and adaptation, all of which resemble iterative optimization. Precisely what algorithms enable the brain to approach optimal performance is not known, however a clue is provided by another prominent idea: compositionality. For about a century, researchers have been talking about motor synergies or primitives which somehow simplify control [9–11]. The implied reduction in dimensionality is now well documented [12–14]. However the structure and origin of the hypothetical primitives, the rules for combining them, and the ways in which they actually simplify the control problem remain unclear.

## 2   Stochastic optimal control problems with linear Bellman equations

We will be able to derive compositionality rules for first-exit and finite-horizon stochastic optimal control problems which belong to a certain class. This class includes both discrete-time [15–17] and continuous-time [17–19] formulations, and is rather general, yet affords substantial simplification. Most notably the optimal control law is found analytically given the optimal cost-to-go, which in turn is the solution to a linear equation obtained from the Bellman equation by exponentiation. Linearity implies compositionality as will be shown here. It also makes a number of other things possible: finding the most likely trajectories of optimally-controlled stochastic systems via deterministic methods; solving inverse optimal control problems via convex optimization; applying off-policy learning in the state space as opposed to the state-action space; establishing duality between stochastic optimal control and Bayesian estimation. An overview can be found in [17]. Here we only provide the background needed for the present paper.

The discrete-time problem is defined by a state cost $q(x) \geq 0$ describing how (un)desirable different states are, and passive dynamics $x' \sim p(\cdot|x)$ characterizing the behavior of the system in the absence of controls. The controller can impose any dynamics $x' \sim u(\cdot|x)$ it wishes, however it pays a price (control cost) which is the KL divergence between $u$ and $p$. We further require that $u(x'|x) = 0$ whenever $p(x'|x) = 0$ so that KL divergence is well-defined. Thus the discrete-time problem is

$$
\begin{aligned}
&\text{dynamics:} && x' \sim u(\cdot|x) \\
&\text{cost rate:} && \ell(x, u(\cdot|x)) = q(x) + KL(u(\cdot|x)\,||\,p(\cdot|x))
\end{aligned}
$$

Let $\mathcal{I}$ denote the set of interior states and $\mathcal{B}$ the set of boundary states, and let $f(x) \geq 0$, $x \in \mathcal{B}$ be a final cost. Let $v(x)$ denote the optimal cost-to-go, and define the *desirability* function

$$
z(x) = \exp(-v(x))
$$

Let $\mathcal{G}$ denote the linear operator which computes expectation under the passive dynamics:

$$
\mathcal{G}[z](x) = E_{x' \sim p(\cdot|x)} z(x')
$$

For $x \in \mathcal{I}$ it can be shown that the optimal control law $u^*(\cdot|x)$ and the desirability $z(x)$ satisfy

$$
\begin{aligned}
&\text{optimal control law:} && u^*(x'|x) = \frac{p(x'|x)\,z(x')}{\mathcal{G}[z](x)} \\
&\text{linear Bellman equation:} && \exp(q(x))\,z(x) = \mathcal{G}[z](x)
\end{aligned}
\tag{1}
$$

On the boundary $x \in \mathcal{B}$ we have $z(x) = \exp(-f(x))$. The linear Bellman equation can be written more explicitly in vector-matrix notation as

$$
\mathbf{z}_{\mathcal{I}} = M\mathbf{z}_{\mathcal{I}} + N\mathbf{z}_{\mathcal{B}}
\tag{2}
$$

where $M = \mathrm{diag}(\exp(-\mathbf{q}_{\mathcal{I}}))\,P_{\mathcal{I}\mathcal{I}}$ and $N = \mathrm{diag}(\exp(-\mathbf{q}_{\mathcal{I}}))\,P_{\mathcal{I}\mathcal{B}}$. The matrix $M$ is guaranteed to have spectral radius less than 1, thus the simple iterative solver $\mathbf{z}_{\mathcal{I}} \leftarrow M\mathbf{z}_{\mathcal{I}} + N\mathbf{z}_{\mathcal{B}}$ converges.

The continuous-time problem is a control-affine Ito diffusion with control-quadratic cost:

$$
\begin{aligned}
&\text{dynamics:} && d\mathbf{x} = \mathbf{a}(\mathbf{x})\,dt + B(\mathbf{x})(\mathbf{u}dt + \sigma d\omega) \\
&\text{cost rate:} && \ell(\mathbf{x}, \mathbf{u}) = q(\mathbf{x}) + \frac{1}{2\sigma^2}\|\mathbf{u}\|^2
\end{aligned}
$$

The control $\mathbf{u}$ is now a (more traditional) vector and $\omega$ is a Brownian motion process. Note that the control cost scaling by $\sigma^{-2}$, which is needed to make the math work, can be compensated by rescaling $q$. The optimal control law $\mathbf{u}^*(\mathbf{x})$ and desirability $z(\mathbf{x})$ satisfy

$$
\begin{aligned}
&\text{optimal control law:} && \mathbf{u}^*(\mathbf{x}) = \sigma^2 B(\mathbf{x})^{\mathsf{T}} \frac{z_{\mathbf{x}}(\mathbf{x})}{z(\mathbf{x})} \\
&\text{linear HJB equation:} && q(\mathbf{x})\,z(\mathbf{x}) = \mathcal{L}[z](\mathbf{x})
\end{aligned}
\tag{3}
$$

where the 2nd-order linear differential operator $\mathcal{L}$ is defined as

$$
\mathcal{L}[z](\mathbf{x}) = \mathbf{a}(\mathbf{x})^{\mathsf{T}} z_{\mathbf{x}}(\mathbf{x}) + \frac{\sigma^2}{2}\,\mathrm{tr}\left(B(\mathbf{x})B(\mathbf{x})^{\mathsf{T}} z_{\mathbf{xx}}(\mathbf{x})\right)
$$

The relationship between the two formulations above is not obvious, but nevertheless it can be shown that the continuous-time formulation is a special case of the discrete-time formulation. This is done by defining the passive dynamics $p_{(h)}(\cdot|\mathbf{x})$ as the $h$-step transition probability density of the uncontrolled diffusion (or an Euler approximation to it), and the state cost as $q_{(h)}(\mathbf{x}) = hq(\mathbf{x})$. Then, in the limit $h \to 0$, the integral equation $\exp(q_{(h)}) z = \mathcal{G}_{(h)}[z]$ reduces to the differential equation $qz = \mathcal{L}[z]$. Note that for small $h$ the density $p_{(h)}(\cdot|\mathbf{x})$ is close to Gaussian. From the formula for KL divergence between Gaussians, the KL control cost in the discrete-time formulation reduces to the quadratic control cost in the continuous-time formulation.

The reason for working with both formulations and emphasizing the relationship between them is that most problems of practical interest are continuous in time and space, yet the discrete-time formulation is easier to work with. Furthermore it leads to better numerical stability because integral equations are better behaved than differential equations. Note also that the discrete-time formulation can be used in both discrete and continuous state spaces, although the latter require function approximation in order to solve the linear Bellman equation [20].

## 3 Compositionality theory

The compositionality developed in this section follows from the linearity of equations (1, 3). We focus on first-exit problems which are more general. An example involving a finite-horizon problem will be given later. Consider a collection of $K$ optimal control problems in our class which all have the same dynamics – $p(\cdot|\mathbf{x})$ in discrete time or $\mathbf{a}(\mathbf{x}), B(\mathbf{x}), \sigma$ in continuous time – the same state cost rate $q(\mathbf{x})$ and the same sets $\mathcal{I}$ and $\mathcal{B}$ of interior and boundary states. These problems differ only in their final costs $f_k(\mathbf{x})$. Let $z_k(\mathbf{x})$ denote the desirability function for problem $k$, and $u_k^*(\cdot|\mathbf{x})$ or $\mathbf{u}_k^*(\mathbf{x})$ the corresponding optimal control law. The latter will serve as primitives for constructing optimal control laws for new problems in our class. We will call the $K$ problems we started with *component* and the new problem *composite*.

Suppose the final cost for the composite problem is $f(\mathbf{x})$, and there exist weights $w_k$ such that

$$f(\mathbf{x}) = -\log\left(\sum_{k=1}^{K} w_k \exp(-f_k(\mathbf{x}))\right) \tag{4}$$

Thus the functions $f_k(\mathbf{x})$ define a $K$-dimensional manifold of composite problems. The above condition ensures that for all boundary/terminal states $\mathbf{x} \in \mathcal{B}$ we have

$$z(\mathbf{x}) = \sum_{k=1}^{K} w_k z_k(\mathbf{x}) \tag{5}$$

Since $z$ is the solution to a linear equation, if (5) holds on the boundary then it must hold everywhere. Thus the desirability function for the composite problem is a linear combination of the desirability functions for the component problems. The weights in this linear combination can be interpreted as compatibilities between the control objectives in the component problems and the control objective in the composite problem. The optimal control law for the composite problem is given by (1, 3).

The above construction implies that both $z$ and $z_k$ are everywhere positive. Since $z$ is defined as an exponent, it must be positive. However this is not necessary for the components. Indeed if

$$f(\mathbf{x}) = -\log\left(\sum_{k=1}^{K} w_k z_k(\mathbf{x})\right) \tag{6}$$

holds for all $\mathbf{x} \in \mathcal{B}$, then (5) and $z(\mathbf{x}) > 0$ hold everywhere even if $z_k(\mathbf{x}) \le 0$ for some $k$ and $\mathbf{x}$. In this case the $z_k$'s are no longer desirability functions for well-defined optimal control problems. Nevertheless we can think of them as generalized desirability functions with similar meaning: the larger $z_k(\mathbf{x})$ is the more compatible state $\mathbf{x}$ is with the agenda of component $k$.

### 3.1 Compositionality of discrete-time control laws

When $z_k(\mathbf{x}) > 0$ the composite control law $u^*$ can be expressed as a state-dependent convex combination of the component control laws $u_k^*$. Combining (5, 1) and using the linearity of $\mathcal{G}$,

$$u^*(\mathbf{x}'|\mathbf{x}) = \sum_k \frac{w_k \mathcal{G}[z_k](\mathbf{x})}{\sum_s w_s \mathcal{G}[z_s](\mathbf{x})} \frac{p(\mathbf{x}'|\mathbf{x}) z_k(\mathbf{x}')}{\mathcal{G}[z_k](\mathbf{x})}$$

The second term above is $u_k^*$. The first term is a state-dependent mixture weight which we denote $m_k(\mathbf{x})$. The composition rule for optimal control laws is then

$$u^*(\cdot|\mathbf{x}) = \sum_k m_k(\mathbf{x}) u_k^*(\cdot|\mathbf{x}) \tag{7}$$

Using the fact that $z_k(\mathbf{x})$ satisfies the linear Bellman equation (1) and $q(\mathbf{x})$ does not depend on $k$, the mixture weights can be simplified as

$$m_k(\mathbf{x}) = \frac{w_k \mathcal{G}[z_k](\mathbf{x})}{\sum_s w_s \mathcal{G}[z_s](\mathbf{x})} = \frac{w_k z_k(\mathbf{x})}{\sum_s w_s z_s(\mathbf{x})} \tag{8}$$

Note that $\sum_k m_k(\mathbf{x}) = 1$ and $m_k(\mathbf{x}) > 0$.

## 3.2 Compositionality of continuous-time control laws

Substituting (5) in (3) and assuming $z_k(\mathbf{x}) > 0$, the control law given by (3) can be written as

$$\mathbf{u}^*(\mathbf{x}) = \sum_k \frac{w_k z_k(\mathbf{x})}{\sum_s w_s z_s(\mathbf{x})} \left[ \frac{\sigma^2}{z_k(\mathbf{x})} B(\mathbf{x})^\mathsf{T} \frac{\partial}{\partial \mathbf{x}} z_k(\mathbf{x}) \right]$$

The term in brackets is $\mathbf{u}_k^*(\mathbf{x})$. We denote the first term with $m_k(\mathbf{x})$ as before:

$$m_k(\mathbf{x}) = \frac{w_k z_k(\mathbf{x})}{\sum_s w_s z_s(\mathbf{x})}$$

Then the composite optimal control law is

$$\mathbf{u}^*(\mathbf{x}) = \sum_k m_k(\mathbf{x}) \mathbf{u}_k^*(\mathbf{x}) \tag{9}$$

Note the similarity between the discrete-time result (7) and the continuous-time result (9), as well as the fact that the mixing weights are computed in the same way. This is surprising given that in one case the control law directly specifies the probability distribution over next states, while in the other case the control law shifts the mean of the distribution given by the passive dynamics.

# 4 Analytical solutions to linear-Gaussian problems with non-quadratic costs

Here we specialize the above results to the case when the components are continuous-time linear quadratic Gaussian (LQG) problems of the form

$$\text{dynamics:} \qquad d\mathbf{x} = A\mathbf{x}dt + B(\mathbf{u}dt + \sigma d\omega)$$

$$\text{cost rate:} \qquad \ell(\mathbf{x}, \mathbf{u}) = \frac{1}{2}\mathbf{x}^\mathsf{T} Q \mathbf{x} + \frac{1}{2\sigma^2} \|\mathbf{u}\|^2$$

The component final costs are quadratic:

$$f_k(\mathbf{x}) = \frac{1}{2}\mathbf{x}^\mathsf{T} F_k \mathbf{x}$$

The optimal cost-to-go function for LQG problems is known to be quadratic [21] in the form

$$v_k(\mathbf{x}, t) = \frac{1}{2}\mathbf{x}^\mathsf{T} V_k(t) \mathbf{x} + \alpha_k(t)$$

At the predefined final time $T$ we have $V_k(T) = F_k$ and $\alpha_k(T) = 0$. The optimal control law is

$$\mathbf{u}_k^*(\mathbf{x}, t) = -\sigma^2 B^\mathsf{T} V_k(t) \mathbf{x}$$

The quantities $V_k(t)$ and $\alpha_k(t)$ can be computed by integrating backward in time the ODEs

$$-\dot{V}_k = Q + A^\mathsf{T} V_k + V_k A^\mathsf{T} - V_k \Sigma V_k \tag{10}$$

$$-\dot{\alpha}_k = \frac{1}{2}\operatorname{tr}(\Sigma V_k)$$

Now consider a composite problem with final cost

$$f(\mathbf{x}) = -\log\left(\sum_k w_k \exp\left(-\frac{1}{2}\mathbf{x}^\mathsf{T} F_k \mathbf{x}\right)\right)$$

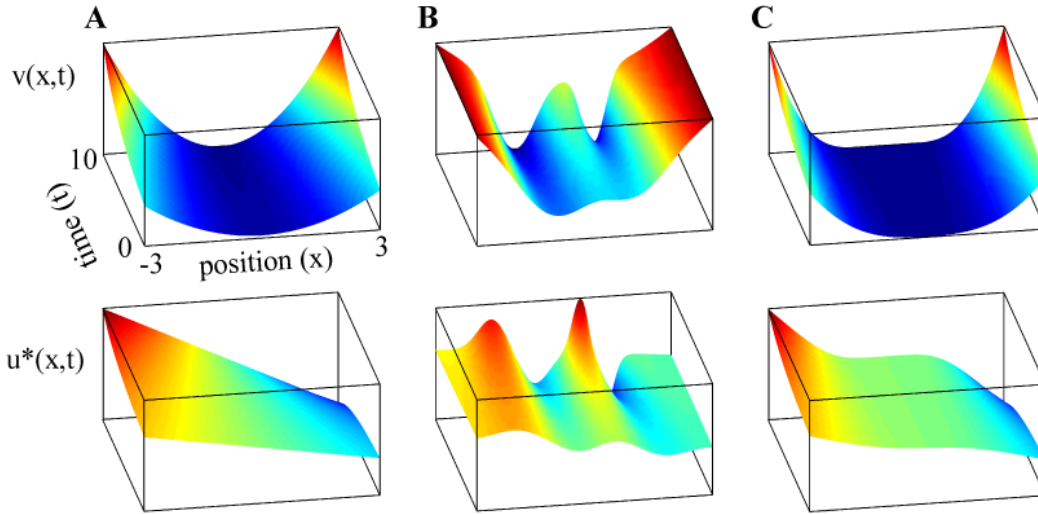

Figure 1: Illustration of compositionality in the LQG framework. (**A**) An LQG problem with quadratic cost-to-go and linear feedback control law. $T = 10$ is the final time. (**B, C**) Non-LQG problems solved analytically by mixing the solutions to multiple LQG problems.

This composite problem is no longer LQG because it has non-quadratic final cost (i.e. log of mixture of Gaussians), and yet we will be able to find a closed-form solution by combining multiple LQG controllers. Note that, since mixtures of Gaussians are universal function approximators, we can represent any desired final cost to within arbitrary accuracy given enough LQG components. Applying the results from the previous section, the desirability for the composite problem is

$$z\left(\mathbf{x}, t\right) = \sum_k w_k \exp\left(-\frac{1}{2}\mathbf{x}^\mathsf{T} V_k\left(t\right)\mathbf{x} - \alpha_k\left(t\right)\right)$$

The optimal control law can now be obtained directly from (3), or via composition from (9). Note that the constants $\alpha_k\left(t\right)$ do not affect the component control laws (and indeed are rarely computed in the LQG framework) however they affect the composite control law through the mixing weights.

We illustrate the above construction on a scalar example with integrator dynamics $dx = udt + 0.2d\omega$. The state cost rate is $q\left(x\right) = 0$. We set $w_k = 1$ for all $k$. The final time is $T = 10$. The component final costs are of the form

$$f_k\left(x\right) = \frac{d_k}{2}\left(x - c_k\right)^2$$

In order to center these quadratics at $c_k$ rather than $0$ we augment the state: $\mathbf{x} = [x; 1]$. The matrices defining the problem are then

$$A = \left[\begin{array}{cc} 0 & 0 \\ 0 & 0 \end{array}\right], \ B = \left[\begin{array}{c} 1 \\ 0 \end{array}\right], \ F_k = d_k \left[\begin{array}{cc} 1 & -c_k \\ -c_k & c_k^2 \end{array}\right]$$

The ODEs (10) are integrated using ode45 in Matlab. **Fig 1** shows the optimal cost-to-go functions $v\left(x, t\right) = -\log\left(z\left(x, t\right)\right)$ and the optimal control laws $u^*\left(x, t\right)$ for the following problems: $\{c = 0; d = 5\}$, $\{c = -1, 0, 1; d = 5, 0.1, 15\}$, and $\{c = -1.5 : 0.5 : 1.5; d = 5\}$. The first problem (**Fig 1A**) is just an LQG. As expected the cost-to-go is quadratic and the control law is linear with time-varying gain. The second problem (**Fig 1B**) has a multimodal cost-to-go. The control law is no longer linear but instead has an elaborate shape. The third problem (**Fig 1C**) resembles robust control in the sense that there is a flat region where all states are equally good. The corresponding control law uses feedback to push the state into this flat region. Inside the region the controller does nothing, so as to save energy. As these examples illustrate, the methodology developed here significantly extends the LQG framework while preserving its tractability.

# 5 Constructing minimal sets of primitives via SVD of Green's function

We showed how composite problems can be solved once the solutions to the component problems are available. The choice of component boundary conditions defines the manifold (6) of problems that can be solved exactly. One can use any available set of solutions as components, but is there a set which is in some sense minimal? Here we offer an answer based on singular value decomposition (SVD). We focus on discrete state spaces; continuous spaces can be discretized following [22].

Recall that the vector of desirability values $z(x)$ at interior states $x \in \mathcal{I}$, which we denoted $\mathbf{z}_\mathcal{I}$, satisfies the linear equation (2). We can write the solution to that equation explicitly as

$$\mathbf{z}_\mathcal{I} = G\, \mathbf{z}_\mathcal{B}$$

where $G = \left(\mathrm{diag}\left(\exp\left(\mathbf{q}_\mathcal{I}\right)\right) - P_{\mathcal{I}\mathcal{I}}\right)^{-1} P_{\mathcal{I}\mathcal{B}}$. The matrix $G$ maps values on the boundary to values on the interior, and thus resembles Green's function for linear PDEs. A minimal set of primitives corresponds to the best low-rank approximation to $G$. If we define "best" in terms of least squares, a minimal set of $R$ primitives is obtained by approximating $G$ using the top $R$ singular values:

$$G \approx USV^\mathsf{T}$$

$S$ is an $R$-by-$R$ diagonal matrix, $U$ and $V$ are $|\mathcal{I}|$-by-$R$ and $|\mathcal{B}|$-by-$R$ orthonormal matrices. If we now set $\mathbf{z}_\mathcal{B} = V_{.r}$, which is the $r$-th column of $V$, then

$$\mathbf{z}_\mathcal{I} = G\, \mathbf{z}_\mathcal{B} \approx USV^\mathsf{T}V_{.r} = S_{rr}U_{.r}$$

Thus the right singular vectors (columns of $V$) are the component boundary conditions, while the left singular vectors (columns of $U$) are the component solutions.

The above construction does not use knowledge of the family of composite problems we aim to solve/approximate. A slight modification makes it possible to incorporate such knowledge. Let the family in question have parametric final costs $f(x,\theta)$. Choose a discrete set $\{\theta_k\}_{k=1\ldots K}$ of values of the parameter $\theta$, and form the $|\mathcal{B}|$-by-$K$ matrix $\Phi$ with elements $\Phi_{ik} = \exp\left(-f\left(x_i, \theta_k\right)\right)$, $x_i \in \mathcal{B}$. As in (4), this choice restricts the boundary conditions that can be represented to $\mathbf{z}_\mathcal{B} = \Phi\mathbf{w}$, where $\mathbf{w}$ is a $K$-dimensional vector. Now apply SVD to obtain a rank-$R$ approximation to the matrix $G\Phi$ instead of $G$. We can set $R \ll K$ to achieve significant reduction in the number of components. Note that $G\Phi$ is smaller than $G$ so the SVD here is faster to compute.

We illustrate the above approach using a discretization of the following 2D problem:

$$\mathbf{a}(\mathbf{x}) = \left[\begin{array}{c} -0.2\,x_2 \\ 0.2\,|x_1| \end{array}\right], \quad B = I, \quad \sigma = 1, \quad q(\mathbf{x}) = 0.1$$

The vector field in **Fig 2A** illustrates the function $\mathbf{a}(\mathbf{x})$. To make the problem more interesting we introduce an L-shaped obstacle which can be hit without penalty but cannot be penetrated. The domain is a disk centered at $(0,0)$ with radius $\sqrt{21}$. The constant $q$ implements a penalty for the time spent inside the disk. The discretization involves $|\mathcal{I}| = 24520$ interior states and $|\mathcal{B}| = 4163$ boundary states. The parametric family of final costs is

$$f(\mathbf{x}, \theta) = 13 - 13\exp\left(5\cos\left(\mathrm{atan2}\left(x_2, x_1\right) - \theta\right) - 5\right)$$

This is an inverted von Mises function specifying the desired location where the state should exit the disk. $f(\mathbf{x}, 0)$ is plotted in red in **Fig 2A**. The set $\{\theta_k\}$ includes 200 uniformly spaced values of $\theta$. The SVD components are constructed using the second method above (although the first method gives very similar results). **Fig 2B** compares the solution obtained with a direct solver (i.e. using the exact $G$) for $\theta = 0$, and the solutions obtained using $R = 70$ and $R = 40$ components. The desirability function $z$ is well approximated in both cases. In fact the approximation to $z$ looks perfect with much fewer components (not shown). However $v = -\log(z)$ is more difficult to approximate. The difficulty comes from the fact that the components are not always positive, and as a result the composite solution is not always positive. The regions where that happens are shown in white in **Fig 2B**. In those regions the approximation is undefined. Note that this occurs only near the boundary. **Fig 2C** shows the first 10 components. They resemble harmonic functions. It is notable that the higher-order components (corresponding to smaller singular values) are only modulated near the boundary – which explains why the approximation errors in **Fig 2B** are near the boundary. In summary, a small number of components are sufficient to construct composite control laws which are near-optimal in most of the state space. Accuracy at the boundary requires additional components. Alternatively one could use positive SVD and obtain not just positive but also more localized components (as we have done in preliminary work).

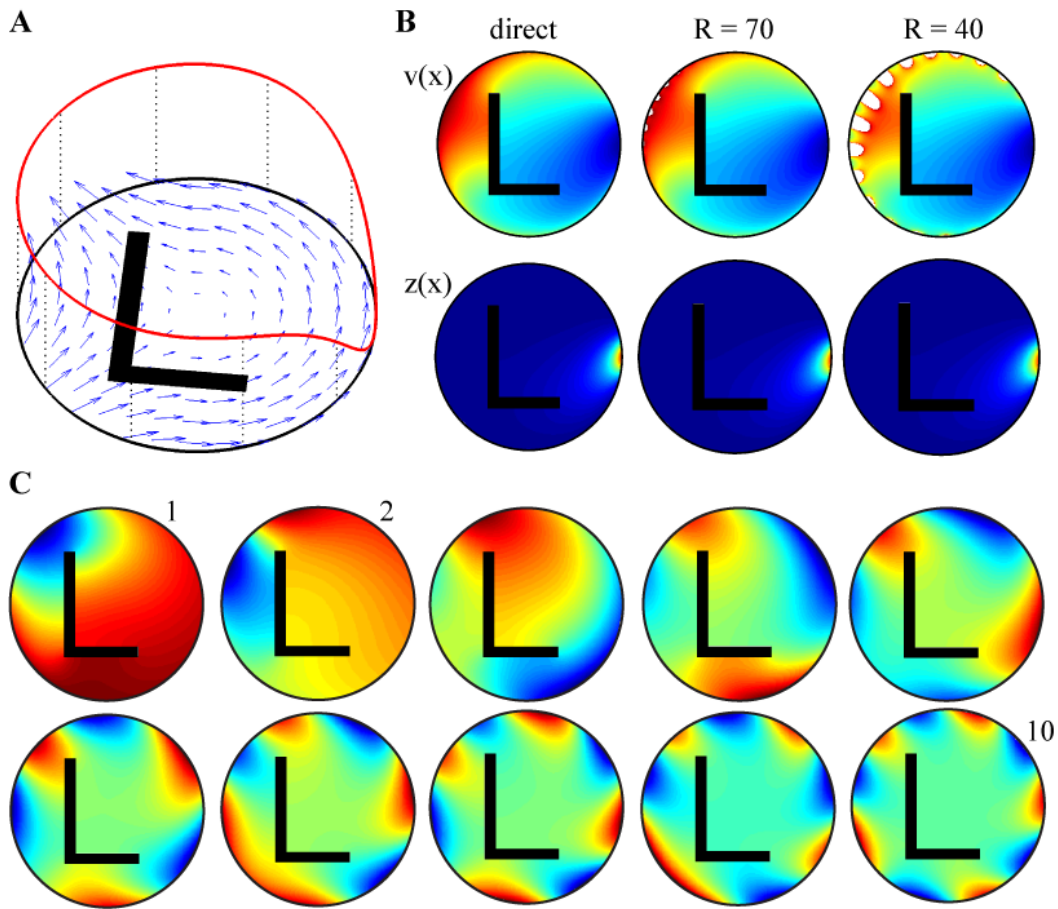

Figure 2: Illustration of primitives obtained via SVD. (**A**) Passive dynamics and cost. (**B**) Solutions obtained with a direct solver and with different numbers of primitives. (**C**) Top ten primitives $z_k(x)$.

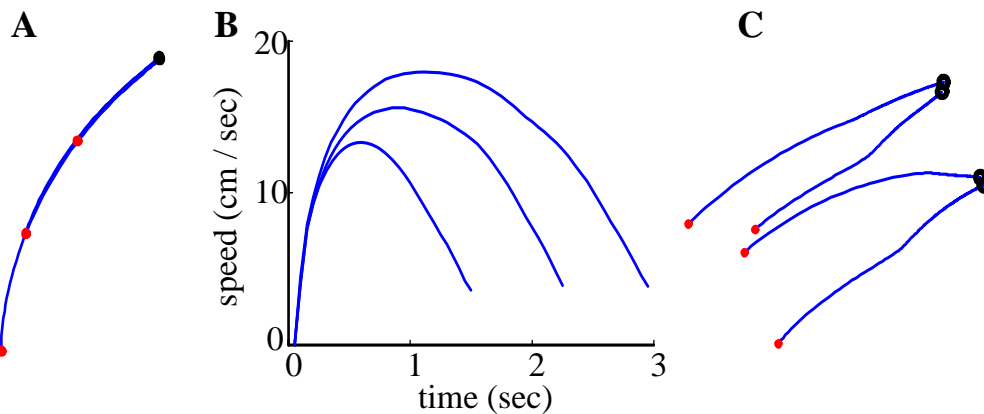

Figure 3: Preliminary model of arm movements. (**A**) Hand paths of different lengths. Red dots denote start points, black circles denote end points. (**B**) Speed profiles for the movements shown in (A). Note that the same controller generates movements of different duration. (**C**) Hand paths generated by a composite controller obtained by mixing the optimal controllers for two targets. This controller "decides" online which target to go to.

# 6   Application to arm movements

We are currently working on an optimal control model of arm movements based on compositionality. The dynamics correspond to a 2-link arm moving in the horizontal plane, and have the form

$$\tau = M\left(\theta\right)\ddot{\theta} + \mathbf{n}\left(\theta, \dot{\theta}\right)$$

$\theta$ contains the shoulder and elbow joint angles, $\tau$ is the applied torque, $M$ is the configuration-dependent inertia, and $\mathbf{n}$ is the vector of Coriolis, centripetal and viscous forces. Model parameters are taken from the biomechanics literature. The final cost $f$ is a quadratic (in Cartesian space) centered at the target. The running state cost is $q = const$ encoding a penalty for duration. The above model has a 4-dimensional state space $(\theta, \dot{\theta})$. In order to encode reaching movements, we introduce an additional state variable $s$ which keeps track of how long the hand speed (in Cartesian space) has remained below a threshold. When $s$ becomes sufficiently large the movement ends. This augmentation is needed in order to express reaching movements as a first-exit problem. Without it the movement would stop whenever the instantaneous speed becomes zero – which can happen at reversal points as well as the starting point. Note that most models of reaching movements have assumed predefined final time. However this is unrealistic because we know that movement duration scales with distance, and furthermore such scaling takes place online (i.e. movement duration increases if the target is perturbed during the movement).

The above second-order system is expressed in general first-order form, and then the passive dynamics corresponding to $\tau = 0$ are discretized in space and time. The time step is $h = 0.02\,\mathrm{sec}$. The space discretization uses a grid with $51^4$x3 points. The factor of 3 is needed to discretize the variable $s$. Thus we have around 20 million discrete states, and the matrix $P$ characterizing the passive dynamics is 20 million - by - 20 million. Fortunately it is very sparse because the noise (in torque space) cannot have a large effect within a single time step: there are about $50$ non-zero entries in each row. Our simple iterative solver converges in about $30$ iterations and takes less than 2 min of CPU time, using custom multi-threaded C++ code.

**Fig 3A** shows hand paths from different starting points to the same target. The speed profiles for these movements are shown in **Fig 3B**. The scaling with amplitude looks quite realistic. In particular, it is known that human reaching movements of different amplitude have similar speed profiles around movement onset, and diverge later. **Fig 3C** shows results for a composite controller obtained by mixing the optimal control laws for two different targets. In this example the targets are sufficiently far away and the final costs are sufficiently steep, thus the mixing yields a switching controller instead of an interpolating controller. Depending on the starting point, this controller takes the hand to one or the other target, and can also switch online if the hand is perturbed. An interpolating controller can be created by placing the targets closer or making the component final costs less steep. While these results are preliminary we find them encouraging. In future work we will explore this model in more detail and also build a more realistic model using 3rd-order dynamics (incorporating muscle time constants). We do not expect to be able to discretize the latter system, but we are in the process of making a transition from discretization to function approximation [20].

# 7   Summary and relation to prior work

We developed a theory of compositionality applicable to a general class of stochastic optimal control problems. Although in this paper we used simple examples, the potential of such compositionality to tackle complex control problems seems clear.

Our work is somewhat related to proto value functions (PVFs) which are eigenfunctions of the Laplacian [5], i.e. the matrix $I - P_{\mathcal{II}}$. While the motivation is similar, PVFs are based on intuitions (mostly from grid worlds divided into rooms) rather than mathematical results regarding optimality of the composite solution. In fact our work suggests that PVFs should perhaps be used to approximate the exponent of the value function instead of the value function itself. Another difference is that PVFs do not take into account the cost rate $q$ and the boundary $\mathcal{B}$. This sounds like a good thing but it may be too good, in the sense that such generality may be the reason why guarantees regarding PVF optimality are lacking. Nevertheless the ambitious agenda behind PVFs is certainly worth pursuing, and it will be interesting to compare the two approaches in more detail.

Finally, another group [6] has developed similar ideas independently and in parallel. Although their paper is restricted to combination of LQG controllers for finite-horizon problems, it contains very interesting examples from complex tasks such as walking, jumping and diving. A particularly important point made by [6] is that the primitives can be only approximately optimal (in this case obtained via local LQG approximations), and yet their combination still produces good results.

## References

[1]  D. Bertsekas, *Dynamic Programming and Optimal Control (2nd Ed)*. Bellmont, MA: Athena Scientific, 2001.

[2]  R. Sutton and A. Barto, *Reinforcement Learning: An Introduction*. MIT Press, Cambridge MA, 1998.

[3]  D. Bertsekas and J. Tsitsiklis, *Neuro-dynamic programming*. Belmont, MA: Athena Scientific, 1997.

[4]  J. Si, A. Barto, W. Powell, and D. Wunsch, *Handbook of Learning and Approximate Dynamic Programming*. Wiley-IEEE Press, 2004.

[5]  S. Mahadevan and M. Maggioni, "Proto-value functions: A Laplacian farmework for learning representation and control in Markov decision processes," *Journal of Machine Learning Research*, vol. 8, pp. 2169–2231, 2007.

[6]  M. daSilva, F. Durand, and J. Popovic, "Linear bellman combination for control of character animation," *To appear in SIGGRAPH*, 2009.

[7]  E. Todorov, "Optimality principles in sensorimotor control," *Nature Neuroscience*, vol. 7, no. 9, pp. 907–915, 2004.

[8]  C. Harris and D. Wolpert, "Signal-dependent noise determines motor planning," *Nature*, vol. 394, pp. 780–784, 1998.

[9]  C. Sherrington, *The integrative action of the nervous system*. New Haven: Yale University Press, 1906.

[10] N. Bernstein, *On the construction of movements*. Moscow: Medgiz, 1947.

[11] M. Latash, "On the evolution of the notion of synergy," in *Motor Control, Today and Tomorrow*, G. Gantchev, S. Mori, and J. Massion, Eds. Sofia: Academic Publishing House "Prof. M. Drinov", 1999, pp. 181–196.

[12] M. Tresch, P. Saltiel, and E. Bizzi, "The construction of movement by the spinal cord," *Nature Neuroscience*, vol. 2, no. 2, pp. 162–167, 1999.

[13] A. D'Avella, P. Saltiel, and E. Bizzi, "Combinations of muscle synergies in the construction of a natural motor behavior," *Nat.Neurosci.*, vol. 6, no. 3, pp. 300–308, 2003.

[14] M. Santello, M. Flanders, and J. Soechting, "Postural hand synergies for tool use," *J Neurosci*, vol. 18, no. 23, pp. 10 105–15, 1998.

[15] E. Todorov, "Linearly-solvable Markov decision problems," *Advances in Neural Information Processing Systems*, 2006.

[16] ——, "General duality between optimal control and estimation," *IEEE Conference on Decision and Control*, 2008.

[17] ——, "Efficient computation of optimal actions," *PNAS, in press*, 2009.

[18] S. Mitter and N. Newton, "A variational approach to nonlinear estimation," *SIAM J Control Opt*, vol. 42, pp. 1813–1833, 2003.

[19] H. Kappen, "Linear theory for control of nonlinear stochastic systems," *Physical Review Letters*, vol. 95, 2005.

[20] E. Todorov, "Eigen-function approximation methods for linearly-solvable optimal control problems," *IEEE International Symposium on Adaptive Dynamic Programming and Reinforcemenet Learning*, 2009.

[21] R. Stengel, *Optimal Control and Estimation*. New York: Dover, 1994.

[22] H. Kushner and P. Dupuis, *Numerical Methods for Stochastic Optimal Control Problems in Continuous Time*. New York: Springer, 2001.

